# Optimal Web-scale Tiering as a Flow Problem

**Gilbert Leung**
eBay, Inc.
San Jose, CA, USA
gleung@alum.mit.edu

**Novi Quadrianto**
SML-NICTA & RSISE-ANU
Canberra, ACT, Australia
novi.quad@gmail.com

**Alexander J. Smola**
Yahoo! Research
Santa Clara, CA, USA
alex@smola.org

**Kostas Tsioutsiouliklis**
Yahoo! Labs
Sunnyvale, CA, USA
kostas@yahoo-inc.com

## Abstract

We present a fast online solver for large scale parametric max-flow problems as they occur in portfolio optimization, inventory management, computer vision, and logistics. Our algorithm solves an integer linear program in an online fashion. It exploits total unimodularity of the constraint matrix and a Lagrangian relaxation to solve the problem as a convex online game. The algorithm generates approximate solutions of max-flow problems by performing stochastic gradient descent on a set of flows. We apply the algorithm to optimize tier arrangement of over 84 million web pages on a layered set of caches to serve an incoming query stream optimally.

## 1 Introduction

Parametric flow problems have been well-studied in operations research [7]. It has received a significant amount of contributions and has been applied in many problem areas such as database record segmentation [2], energy minimization for computer vision [10], critical load factor determination in two-processor systems [16], end-of-session baseball elimination [6], and most recently by [19, 18, 20] in product portfolio selection. In other words, it is a key technique for many estimation and assignment problems. Unfortunately many algorithms proposed in the literature are geared towards thousands to millions of objects rather than billions, as is common in web-scale problems.

Our motivation for solving parametric flow is the problem of webpage tiering for search engine indices. While our methods are entirely general and could be applied to a range of other machine learning and optimization problems, we focus on webpage tiering as the illustrative example in this paper. The rationale for choosing this application is threefold: firstly, it is a real problem in search engines. Secondly, it provides very large datasets. Thirdly, in doing so we introduce a new problem to the machine learning community. That said, our approach would also be readily applicable to very large scale versions of the problems described in [2, 16, 6, 19].

The specific problem that will provide our running example is that of assigning webpages to several tiers of a search engine cache such that the time to serve a query is minimized. For a given query, a search engine returns a number of documents (typically 10). The time it takes to serve a query depends on where the documents are located. The first tier (or cache) is the fastest (using premium hardware, etc. thus also often the smallest) and retrieves its documents with little latency. If even just a single document is located in a back tier, the delay is considerably increased since now we need to search the larger (and slower) tiers until the desired document is found. Hence it is our goal to assign the most popular documents to the fastest tiers while taking the interactions between documents into account.

## 2 The Tiering Problem

We would like to allocate documents $d \in D$ into $k$ tiers of storage at our disposal. Moreover, let $q \in Q$ be the queries arriving at a search engine, with finite values $v_q > 0$ (e.g. the probability of the query, possibly weighted by the relevance of the retrieved results), and a set of documents $D_q$ retrieved for the query. This input structure is stored in a bipartite graph $G$ with vertices $V = D \cup Q$ and edges $(d, q) \in E$ whenever document $d$ should be retrieved for query $q$.

The $k$ tiers, with tier 1 as the most desirable and $k$ the least (most costly for retrieval), form an increasing sequence of *cummulative* capacities $C_t$, with $C_t$ indicating how many pages can be stored by tiers $t' \leq t$ together. Without loss of generality, assume $C_{k-1} < |D|$ (that is, the last tier is required to hold all documents, or the problem can be reduced). Finally, for each $t \geq 2$ we assume that there is a penalty $p_{t-1} > 0$ incurred by a tier-miss at level $t$ (known as "fallthrough" from tier $t - 1$ to tier $t$). And since we have to access tier 1 regardless, we set $p_0 = 0$ for convenience. For instance, retrieving a page in tier 3 incurs a total penalty of $p_1 + p_2$.

### 2.1 Background

Optimization of index structures and data storage is a key problem in building an efficient search engine. Much work has been invested into building efficient inverted indices which are optimized for query processing [17, 3]. These papers all deal with the issue of optimizing the data *representation* for a given query and how an inverted index should be stored and managed for general queries. In particular, [3, 14] address the problem of computing the top-$k$ results without scanning over the entire inverted lists. Recently, machine learning algorithms have been proposed [5] to improve the ordering within a given collection beyond the basic inverted indexing setup [3].

A somewhat orthogonal strategy to this is to decompose the collection of webpages into a number of disjoint tiers [15] ordered in decreasing level of relevance. That is, documents are partitioned according to their relevance for answering queries into different tiers of (typically) increasing size. This leads to putting the most frequently retrieved or the most relevant (according to the value of query, the market or other operational parameters) pages into the top tier with the smallest latency and relegating the less frequently retrieved or the less relevant pages into bottom tiers. Since queries are often carried out by *sequentially* searching this hierarchy of tiers, an improved ordering minimizes latency, improves user satisfaction, and it reduces computation.

A naive implementation of this approach would simply assign a value to each page in the index and arrange them such that the most frequently accessed pages reside in the highest levels of the cache. Unfortunately this approach is suboptimal: in order to answer a given query well a search engine typically does not only return a *single* page as a result but rather returns a *list* of $r$ (typically $r = 10$) pages. This means that if even just one of these pages is found at a much lower tier, we either need to search the backtiers to retrieve this page or alternatively we need to sacrifice result relevance.

At first glance, the problem is daunting: we need to take all correlations among pages induced by user queries into account. Moreover, for reasons of practicality we need to design an algorithm which is linear in the amount of data presented (i.e. the number of queries) and whose storage requirements are only linear in the number of pages. Finally, we would like to obtain guarantees in terms of performance for the assignment that we obtain from the algorithm. Our problem, even for $r = 2$, is closely related to the weighted $k$-densest subgraph problem, which is NP hard [13].

### 2.2 Optimization Problem

Since the problem we study is somewhat more general than the parametric flow problem we give a self-contained derivation of the problem and derive the more general version beyond [7]. For brevity, we relegate all proofs to the Appendix.

We denote the result set for query $q$ by $D_q := \{d : (d, q) \in G\}$, and similarly, the set of queries seeking for a document $d$ by $Q_d := \{q : (d, q) \in G\}$. For a document $d$ we denote by $z_d \in \{1, \dots, k\}$ the tier storing $d$. Define

$$u_q := \max_{d \in D_q} z_d \qquad (1)$$

as the number of cache levels we need to traverse to answer query $q$. In other words, it is the document found in the worst tier which determines the cost of access. Integrating the optimization over $u_q$ we may formulate the tiering problem as an integer program:

$$\underset{z,u}{\text{minimize}} \sum_{q \in Q} v_q \sum_{t=1}^{u_q-1} p_t \text{ subject to } z_d \leq u_q \leq k \text{ for all } (q,d) \in G \text{ and } \sum_{d \in D} \{z_d \leq t\} \leq C_t \ \forall \ t. \tag{2}$$

Note that we replaced the maximization condition (1) by a linear inequality in preparation for a reformulation as an integer linear program. Obviously, the optimal $u_q$ for a given $z$ will satisfy (1).

**Lemma 1** *Assume that $C_k \geq |D| > C_{k-1}$. Then there exists an optimal solution of (2) such that $\sum_d \{z_d \leq t\} = C_t$ for all $1 \leq t < k$.* ∎

In the following we address several issues associated with the optimization problem: A) Eq. (2) is an *integer* program and consequently it is discrete and nonconvex. We show that there exists a convex reformulation of the problem. B) It is at a formidable scale (often $|D| > 10^9$). Section 3.4 presents a stochastic gradient descent procedure to solve the problem in few passes through the database. C) We have insufficient data for an accurate tier assignment for pages associated with tail queries. This can be addressed by a smoothing estimator for the tier index of a page.

## 2.3 Integer Linear Program

We now replace the selector variables $z_d$ and $u_q$ by binary variables via a "thermometer" code. Let

$$x \in \{0;1\}^{D \times (k-1)} \text{ subject to } x_{dt} \geq x_{d,t+1} \text{ for all } d,t \tag{3a}$$

$$y \in \{0;1\}^{Q \times (k-1)} \text{ subject to } y_{qt} \geq y_{q,t+1} \text{ for all } q,t \tag{3b}$$

be index variables. Thus we have the one-to-one mapping $z_d = 1 + \sum_t x_{dt}$ and $x_{dt} = \{z_d > t\}$ between $z$ and $x$. For instance, for $k=5$, a middle tier $z=3$ maps into $x=(1,1,0,0)$ (requiring two fallthroughs), and the best tier $z=1$ corresponds to $x=(0,0,0,0)$. The mapping between $u$ and $y$ is analogous. The constraint $u_q \geq z_d$ can simply be rewritten coordinate-wise $y_{qt} \geq x_{dt}$.

Finally, the capacity constraints assume the form $\sum_d x_{dt} \geq |D| - C_t$. That is, the number of pages allocated to higher tiers are at least $|D| - C_t$. Define *remaining* capacities $\bar{C}_t := |D| - C_t$ and use the variable transformation (1) we have the following integer linear program:

$$\underset{x,y}{\text{minimize}} \ v^\top y p \tag{4a}$$

$$\text{subject to } x_{dt} \geq x_{d,t+1} \text{ and } y_{qt} \geq y_{q,t+1} \text{ and } y_{qt} \geq x_{dt} \text{ for all } (q,d) \in G \tag{4b}$$

$$\sum_d x_{dt} \geq \bar{C}_t \text{ for all } 1 \leq t \leq k-1 \tag{4c}$$

$$x \in \{0;1\}^{D \times (k-1)}; y \in \{0;1\}^{Q \times (k-1)} \tag{4d}$$

where $p = (p_1, \ldots, p_{k-1})^\top$ and $v = (v_1, \ldots, v_{|Q|})^\top$ are column vectors, and $y$ a matrix $(y_{qt})$. The advantage of (4) is that while still discrete, we now have *linear* constraints and a *linear* objective function. The only problem is that the variables $x$ and $y$ need to be binary.

**Lemma 2** *The solutions of (2) and (4) are equivalent.* ∎

## 2.4 Hardness

Before discussing convex relaxations and approximation algorithms it is worthwhile to review the hardness of the problem: consider only two tiers, and a case where we retrieve only *two* pages per query. The corresponding graph has vertices $D$ and edges $(d,d') \in E$, whenever $d$ and $d'$ are displayed together to answer a query. In this case the tiering problem reduces to one of finding a subset of vertices $D' \subset D$ such that the induced subgraph has the largest number (possibly weighted) of edges subject to the capacity constraint $|D'| \leq C$.

For the case of $k$ pages per query, simply assume that $k-2$ of the pages are always the same. Hence the problem of finding the best subset reduces to the case of 2 pages per query. This problem is identical to the $k$-densest subgraph problem which is known to be NP hard [13].

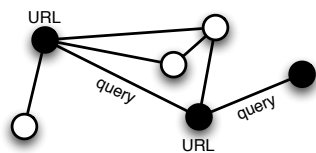

Figure 1: $k$-densest subgraph reduction. Vertices correspond to URLs and queries correspond to edges. Queries can be served whenever the corresponding URLs are in the cache. This is the case whenever the induced subgraph contains the edge.

## 3 Convex Programming

The key idea in solving (4) is to relax the capacity constraints for the tiers. This renders the problem totally unimodular and therefore amenable to a solution by a linear program. We replace the capacity constraint by a partial Lagrangian. This does *not* ensure that we will be able to meet the capacity constraints *exactly* anymore. Instead, we will only be able to state ex-post that the relaxed solution is optimal for the *observed* capacity distribution. Moreover, we are still able to control capacity by a suitable choice of the associated Lagrange multipliers.

### 3.1 Linear Program

Instead of solving (4) we study the linear program:

$$\underset{x,y}{\text{minimize}} \ v^\top y p - \mathbf{1}^\top x \lambda \text{ subject to } x_{dt} \geq x_{d,t+1} \text{ and } y_{qt} \geq y_{q,t+1} \tag{5}$$

$$y_{qt} \geq x_{dt} \text{ for } (q,d) \in G \text{ and } x_{dt}, y_{qt} \in [0,1]$$

Here $\lambda = (\lambda_1, \ldots, \lambda_{k-1})^\top$ act as Lagrange multipliers $\lambda_t \geq 0$ for enforcing capacity constraints and $\mathbf{1}$ denotes a column of $|D|$ ones. We now relate the solution of (5) to that of (4).

**Lemma 3** *For any choice of $\lambda$ with $\lambda_t \geq 0$ the linear program (5) has an integral solution, i.e. there exists some $x^*, y^*$ satisfying $x^*_{dt}, y^*_{qt} \in \{0; 1\}$ which minimize (5). Moreover, for $\bar{C}_t = \sum_d x^*_{dt}$ the solution $(x^*, y^*)$ also solves (4).* ∎

We have succeeded in reducing the complexity of the problem to that of a linear program, yet it is still formidable and it needs to be solved to optimality for an accurate caching prescription. Moreover, we need to adjust $\lambda$ such that we satisfy the desired capacity constraints (approximately).

**Lemma 4** *Denote by $L^*(\lambda)$ the value of (5) at the solution of (5) and let $L(\lambda) := L^*(\lambda) + \sum_t \bar{C}_t \lambda_t$. Hence $L(\lambda)$ is concave in $\lambda$ and moreover, $L(\lambda)$ is maximized for a choice of $\lambda$ where the solution of (5) satisfies the constraints of (4).* ∎

Note that while the above two lemmas provide us with a guarantee that for every $\lambda$ and for every associated integral solution of (5) there exists a set of capacity constraints for which this is optimal and that such a capacity satisfying constraint can be found efficiently by concave maximization, they do not guarantee the converse: not every capacity constraint can be satisfied by the convex relaxation, as the following example demonstrates.

**Example 1** *Consider the case of 2 tiers (hence we drop the index $t$), a single query $q$ and 3 documents $d$. Set the capacity constraint of the first tier to 1. In this case it is impossible to avoid a cache miss in the ILP. In the LP relaxation of (4), however, the optimal (non-integral) solution is to set all $x_d = \frac{1}{3}$ and $y_q = \frac{1}{3}$. The partial Lagrangian $L(\lambda)$ is maximized for $\lambda = -p/3$. Moreover, for $\lambda < -p/3$ the optimization problem (5) has as its solution $x = y = 1$; whereas for $\lambda > -p/3$ the solution is $x = y = 0$. For the critical value any convex combination of those two values is valid.*

This example shows why the optimal tiering problem is NP hard — it is possible to design cases where the tier assignment for a page is highly ambiguous. Note that for the *integer* programming problem with capacity constraint $C = 2$ we could allocate an arbitrary pair of pages to the cache. This does not change the objective function (total cache miss) or feasibility.

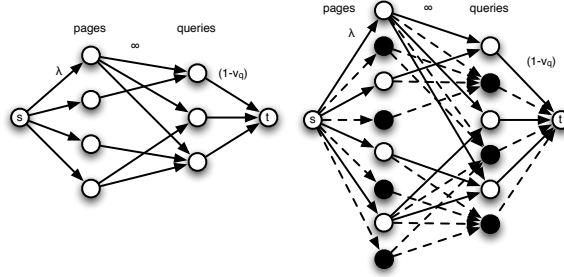

Figure 2: Left: maximum flow problem for a problem of 4 pages and 3 queries. The minimum cut of the directed graph needs to sever all pages leading to a query or alternatively it needs to sever the corresponding query incurring a penalty of $(1 - v_q)$. This is precisely the tiering objective function for the case of two tiers. Right: the same query graph for three tiers. Here the black nodes and dashed edges represent a copy of the original graph — additionally each page in the original graph also has an infinite-capacity link to the corresponding query in the additional graph.

## 3.2 Graph Cut Equivalence

It is well known that the case of two tiers ($k = 2$) can be relaxed to a min-cut, max-flow problem [7, 4]. The transformation works by designing a bipartite graph between queries $q$ and documents $d$. All documents are connected to the source $s$ by edges with capacity $\lambda$ and queries are connected to the sink $t$ with capacity $(1 - v_q)$. Documents $d$ retrieved for a query $q$ are connected to $q$ with capacity $\infty$.

Figure 2 provides an example of such a maximum-flow, minimum-cut graph from source $s$ to sink $t$. The conversion to several tiers is slightly more involved. Denote by $v_{di}$ vertices associated with document $d$ and tier $i$ and moreover, denote by $w_{qi}$ vertices associated with a query $q$ and tier $i$. Then the graph is given by edges $(s, v_{di})$ with capacities $\lambda_i$; edges $(v_{di}, w_{qi'})$ for all (document, query) pairs and for all $i \leq i'$, endowed with infinite capacity; and edges $(w_{qi}, t)$ with capacity $(1 - v_q)$.

As with the simple caching problem, we need to impose a cut on any query edge for which not all incoming page edges have been cut. The key difference is that in order to benefit from storing pages in a better tier we need to guarantee that the page is contained in the lower tier, too.

## 3.3 Variable Reduction

We now simplify the relaxed problem (5) further by reducing the number of variables, without sacrificing integrality of the solution. A first step is to substitute $y_{qt} = \max_{d \in D_q} x_{dt}$, to obtain an optimization problem over the documents alone:

$$\underset{x}{\text{minimize}} \; v^\top \Big( \max_{d \in D_q} x_{dt} \Big) p - 1^\top x \lambda \text{ subject to } x_{dt} \geq x_{dt'} \text{ for } t' > t \text{ and } x_{dt} \in [0, 1] \qquad (6)$$

Note that the monotonicity condition $y_{qt} \geq y_{qt'}$ for $t' > t$ is automatically inherited from that of $x$. The solution of (6) is still integral since the problem is equivalent to one with integral solution.

**Lemma 5** *We may scale $p_t$ and $\lambda_t$ together by constants $\beta_t > 0$, such that $p'_t / p_t = \beta_t = \lambda'_t / \lambda_t$. The resulting solution of this new problem (6) with $(p', \lambda')$ is unchanged.* ∎

Essentially, problem (5) as parameterized by $(p, \lambda)$ yields solutions which form equivalence classes. Consequently for the convenience of solving (5), we may assume $p'_t = 1$ for $t \geq 1$. We only need to consider the original $p$ for evaluating the objective using solution $z$ (thus, same observed capacities $C_t$).

Since (5) is a relaxation of (4) this reformulation can be extended to the integer linear program, too. Moreover, under reasonable conditions on the capacity constraints, there is more structure in $\lambda$.

**Lemma 6** *Assume that $\bar{C}_t$ is monotonically decreasing and that $p_t = 1$ for $t \geq 1$. Then any choice of $\lambda$ satisfying the capacity constraints is monotonically non-increasing.* ∎

| **Algorithm 1** Tiering Optimization | **Algorithm 2** Deferred updates |
|---|---|
| Initialize all $z_d = 0$ <br> Initialize $n = 100$ <br> **for** $i = 1$ **to** MAXITER **do** <br>   **for** all $q \in Q$ **do** <br>     $\eta = \frac{1}{\sqrt{n}}$ (learning rate) <br>     $n \leftarrow n + 1$ (increment counter) <br>     Update $z \leftarrow z - \eta \partial_x \ell_q(z)$ <br>     Project $z$ to $[1, k]^D$ via <br>     $z_d \leftarrow \max(1, \min(k, z_d))$ <br>   **end for** <br> **end for** | Observe current time $n'$ <br> Read timestamp $n$ for document $d$ <br> Compute update steps $\delta = \delta(n', n)$ <br> **repeat** <br>   $j = \lfloor z_d + 1 \rfloor$ (next largest tier) <br>   $t = (j - z_d)/\lambda_j$ (change needed to reach next tier) <br>   **if** $t > \delta$ **then** <br>     $\delta = 0$ and $z_d \leftarrow z_d + \lambda_j \delta$ (partial step; we are done) <br>   **else** <br>     $\delta \leftarrow \delta - t$ and $z_d \leftarrow z_d + 1$ (full step; next tier) <br>   **end if** <br> **until** $\delta = 0$ (no more updates) or $z_d = k-1$ (bottom tier) |

One interpretation of this is that, unless the tiers are increasingly inexpensive, the optimal solution would assign pages in a fashion yielding empty middle tiers (the remaining capacities $\bar{C}_t$ not strictly decreasing). This monotonicity simplifies the problem. Consequently, we exploit this fact to complete the variable reduction.

Define $\delta \lambda_i := \lambda_i - \lambda_{i+1}$ for $i \geq 1$ (all non-negative by virtue of Lemma 6) and

$$f_\lambda(\chi) := -\lambda_1 \chi + \sum_{i=1}^{k-2} \delta \lambda_i \max(0, i - \chi) \text{ for } \chi \in [0, \text{k-1}]. \tag{7}$$

Note that by construction $\partial_\chi f_\lambda(\chi) = -\lambda_i$ whenever $\chi \in (i-1, i)$. The function $f_\lambda$ is clearly convex, which helps describe our tiering problem via the following convex program

$$\underset{z}{\text{minimize}} \, v^\top \left( \max_{d \in D_q} z_d \right) + \sum_d f_\lambda(z_d - 1) \text{ for } z_d \in [1, k] \tag{8}$$

We now use only one variable per document. Moreover, the convex constraints are simple box constraints. This simplifies convex projections, as needed for online programming.

**Lemma 7** *The solution of (8) is equivalent to that of (5).*     ■

### 3.4 Online Algorithm

We now turn our attention to a fast algorithm for minimizing (8). While greatly simplified relative to (2) it still remains a problem of billions of variables. The key observation is that the objective function of (8) can be written as sum over the following loss functions

$$l_q(z) := v_q \max_{d \in D_q} z_d + \frac{1}{|Q|} \sum_d f_\lambda(z_d - 1) \tag{9}$$

where $|Q|$ denotes the cardinality of the query set. The transformation suggests a simple stochastic gradient descent optimization algorithm: traverse the input stream by queries, and update the values of $x_d$ of all those documents $d$ that would need to move into the next tier in order to reduce service time for a query. Subsequently, perform a projection of the page vectors to the set $[1, k]$ to ensure that we do not assign pages to non-existent tiers.

Algorithm 1 proceeds by processing the input query-result records $(q, v_q, D_q)$ as a stream comprising the set of pages that need to be displayed to answer a given query. More specifically, it updates the tier preferences of the pages that have the lowest tier scores for each level and it decrements the preferences for all other pages. We may apply results for online optimization algorithms [1] to show that a small number of passes through the dataset suffice.

**Lemma 8** *The solution obtained by Algorithm 1 converges at rate $O(\sqrt{(\log T)/T})$ to its minimum value. Here $T$ is the number of queries processed.*

### 3.5  Deferred and Approximate Updates

The naive implementation of algorithm 1 is infeasible as it would require us to update all $|D|$ coordinates of $x_d$ for each query $q$. However, it is possible to defer the updates until we need to inspect $z_d$ directly. The key idea is to exploit that for all $z_d$ with $d \notin D_q$ the updates only depend on the value of $z_d$ at update time (Section A.1) and that $f_\lambda$ is piecewise linear and monotonically decreasing.

### 3.6  Path Following

The tiering problem has the appealing property [19] that the solutions for increasing $\lambda$ form a nested subset. In other words, relaxing capacity constraints never demotes but only promotes pages. This fact can be used to design specialized solvers which work well at determining the entire solution path at once for moderate-sized problems [19]. Alternatively, we can simply take advantage of solutions for successive values of $\lambda$ in determining an approximate solution path by using the solution for $\lambda$ as initialization for $\lambda'$. This strategy is well known as path-following in numerical optimization.

In this context it is undesirable to solve the optimization for a particular value of $\lambda$ to optimality. Instead, we simply solve it approximately (using a small number of passes) and readjust $\lambda$. Due to the nesting property [19] and the fact that the optimal solutions are binary (via total unimodularity) the average over solutions on the entire path provides an ordering of pages into tiers. Thus,

**Lemma 9** *Denote by $x_d(\lambda)$ the solution of the two-tier optimization problem for a given value of $\lambda$. Moreover, denote by $\zeta_d := [\lambda' - \lambda]^{-1} \int_\lambda^{\lambda'} x_d(\lambda)$ the average value over a range of Lagrange multipliers. Then $\zeta_d$ provides an order for sorting documents into tiers for the entire range $[\lambda, \lambda']$.*

In practice[1], we only choose a finite number of steps for near-optimal solutions. This yields

---
**Algorithm 3** Path Following

---
Initialize all $(x_{dt}) = z_d \in [1, k]$
**for** each $\lambda \in \Lambda$ **do**
  Refine variables $x_{dt}(\lambda)$ by Algorithm 1 using a small number of iterations.
**end for**
Average the variables $x_{dt} = \sum_{\lambda \in \Lambda} x_{dt}(\lambda)/|\Lambda|$
Sort the documents with the resulting total scores $z_d$
Fill the ordered documents to tier 1, then tier 2, etc.

---

Experiments show that using synthetic data (where it was feasible to compute and compare with the optimal LP solution pointwise) even $|\Lambda| = 5$ values of $\lambda$ produce near-optimal results in the two-tier case. Moreover, we may carry out the optimization procedure for several parameters simultaneously. This is advantageous since the main cost is sequential RAM read-write access rather than CPU speed.

## 4  Experiments

To examine the efficacy of our algorithm at web-scale we tested it with real data from a major search engine. The results of our proposed methods are compared to those of the *max* and *sum* heuristics in Section A.2. We also performed experiments on small synthetic data (2-tier and 3-tier), where we were able to show that our algorithm converges to exact solution given by an LP solver (Appendix C). However, since LP solvers are very slow, it is not feasible for web-scale problems.

We processed the logs for one week of September 2009 containing results from the top geographic regions which include a majority of the search engine's user base. To simplify the heavy processing involved for collecting such a massive data set, we only record whether a particular *result*, defined as a (query, document) pair, appears in top 10 (first result page) for a given session and we aggregate the view counts of such results, which will be used for the session value $v_q$ once. In its entirety this subset contains about $10^8$ viewed documents and $1.6 \cdot 10^7$ distinct queries. We excluded results viewed only once, yielding a final data set of $8.4 \cdot 10^7$ documents.[2] For simplicity, our experiments are carried out for a two-tier (single cache) system such that the only design parameter is the relative

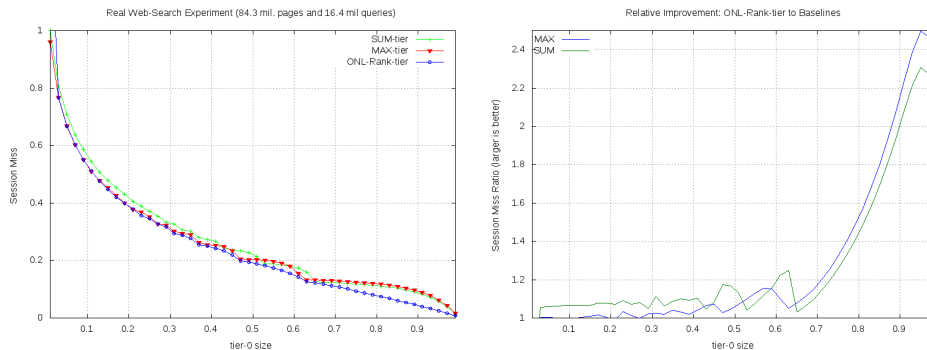

Figure 3: Left: Experimental results for real web-search data with $8.4 \cdot 10^7$ pages and $1.6 \cdot 10^7$ queries. Session miss rate for the online procedure, the *max* and *sum* heuristics (A.2). (The $y$-axis is normalized such that SUM-tier's first point is at 1). As seen, the max heuristic cannot be optimal for any but small cache sizes, but it performs comparably well to Online. Right: "Online" is outperforming MAX for cache size larger than 60%, sometimes more than twofold.

size of the prime tier (the cache). The ranking variant of our online Algorithm 3 (30 passes over the data) consistently outperforms the max and sum heuristics over a large span of cache sizes (Figure 3).

Direct comparison can now be made between our online procedure and the max and sum heuristics since each one induces a ranking on the set of documents. We then calculate the session miss rate of each procedure at any cache size, and report the relative improvement of our online algorithm as ratios of miss rates in Figure 3–Right.

The optimizer fits well in a desktop's RAM since 5 values of $\lambda$ only amount to about 2GB of single-precision $x(\lambda)$. We measure a throughput of approximately 0.5 million query-sessions per second (qps) for this version, and about 2 million qps for smaller problems (as they incur fewer memory page faults). Billion-scale problems can readily fit in 24GB of RAM by serializing computation one $\lambda$ value at a time. We also implemented a multi-thread version utilizing 4 CPU cores, although its performance did not improve since memory and disk bandwidth limits have already been reached.

## 5    Discussion

We showed that very large tiering and densest subset optimization problems can be solved efficiently by a relatively simple online optimization procedure (Some extensions are in Appendix B). It came somewhat as a surprise that the max heuristic often works nearly as well as the optimal tiering solution. Since we experienced this correlation on both synthetic and real data we believe that it might be possible to prove approximation guarantees for this strategy whenever the bipartite graphs satisfy certain power-law properties.

Some readers may question the need for a static tiering solution, given that data could, in theory, be reassigned between different caching tiers on the fly. The problem is that in production systems of a search engine, such reassignment of large amounts of data may not always be efficient for operational reasons (e.g. different versions of the ranking algorithm, different versions of the index, different service levels, constraints on transfer bandwidth). In addition to that, tiering is a problem not restricted to the provision of webpages. It occurs in product portfolio optimization and other resource constrained settings. We showed that it is possible to solve such problems at several orders of magnitude larger scale than what was previously considered feasible.

**Acknowledgments**    We thank Kelvin Fong for providing computer facilities. NICTA is funded by the Australian Government as represented by the Department of Broadband, Communications and the Digital Economy and the Australian Research Council through the ICT Centre of Excellence program. This work was carried out while GL and NQ were with Yahoo! Labs.

---

the optimization problem and keeps the model accurate. Moreover, we remove rare results by maintaining that the lowest count of a document is at least as large as the square root of the highest within the same session.

## Footnotes

[1]This result can be readily extended to $k > 2$, and any probability measure over a set of Lagrangian values $\lambda \in \Lambda \subseteq \mathbb{R}_+^{k-1}$ so long as there are positive weights around the values yielding all the nested solutions.

[2]The search results for any fixed query vary for a variety of reasons, e.g. database updates. We approximate the session graph by treating queries with different result sets as if they were different. This does not change

# References

[1] P. Bartlett, E. Hazan, and A. Rakhlin. Adaptive online gradient descent. In J. C. Platt, D. Koller, Y. Singer, and S. Roweis, editors, *NIPS 20*, Cambridge, MA, 2008.

[2] M. J. Eisner and D. G. Severance. Mathematical techniques for efficient record segmentation in large shared databases. *J. ACM*, 23(4):619–635, 1976.

[3] R. Fagin. Combining fuzzy information from multiple systems. In *Fifteenth ACM SIGACT-SIGMOD-SIGART Symposium on Principles of Database Systems*, pages 216–226, Montreal, Canada, 1996.

[4] L. R. Ford and D. R. Fulkerson. Maximal flow through a network. *Canadian Journal of Mathematics*, 8:399–404, 1956.

[5] S. Goel, J. Langford, and A. Strehl. Predictive indexing for fast search. In D. Koller, D. Schuurmans, Y. Bengio, and L. Bottou, editors, *NIPS*, pages 505–512. MIT Press, 2008.

[6] D. Gusfield and C. U. Martel. A fast algorithm for the generalized parametric minimum cut problem and applications. *Algorithmica*, 7(5&6):499–519, 1992.

[7] D. Gusfield and É. Tardos. A faster parametric minimum-cut algorithm. *Algorithmica*, 11(3):278–290, 1994.

[8] I. Heller and C. Tompkins. An extension of a theorem of dantzig's. In H. Kuhn and A. Tucker, editors, *Linear Inequalities and Related Systems*, volume 38 of *Annals of Mathematics Studies*. AMS, 1956.

[9] J. Kleinberg. Authoritative sources in a hyperlinked environment. *Journal of the ACM*, 46(5):604–632, 1999.

[10] V. Kolmogorov, Y. Boykov and C. Rother. Applications of parametric maxflow in computer vision. *ICCV*, 1–8, 2007.

[11] Y. Nesterov and J.-P. Vial. Confidence level solutions for stochastic programming. Technical Report 2000/13, Université Catholique de Louvain - Center for Operations Research and Economics, 2000.

[12] L. Page, S. Brin, R. Motwani, and T. Winograd. The pagerank citation ranking: Bringing order to the web. Technical report, Stanford Digital Library Technologies Project, Stanford, CA, USA, Nov. 1998.

[13] C. H. Papadimitriou and K. Steiglitz. *Combinatorial Optimization: Algorithms and Complexity*. Prentice-Hall, New Jersey, 1982.

[14] M. Persin, J. Zobel, and R. Sacks-Davis. Filtered document retrieval with frequency-sorted indexes. *JASIS*, 47(10):749–764, 1996.

[15] K. M. Risvik, Y. Aasheim, and M. Lidal. Multi-tier architecture for web search engines. In *LA-WEB*, pages 132–143. IEEE Computer Society, 2003.

[16] H. S. Stone. Critical load factors in two-processor distributed systems. *IEEE Trans. Softw. Eng.*, 4(3):254–258, 1978.

[17] H. Yan, S. Ding, and T. Suel. Inverted index compression and query processing with optimized document ordering. In J. Quemada, G. León, Y. Maarek, and W. Nejdl, editors, *18th International Conference on World Wide Web, Madrid, Spain*, pages 401–410. ACM, 2009.

[18] B. Zhang, J. Ward, and A. Feng. A simultaneous maximum flow algorithm for the selection model. Technical Report HPL-2005-91, Hewlett Packard Laboratories, 2005.

[19] B. Zhang, J. Ward, and Q. Feng. A simultaneous parametric maximum-flow algorithm for finding the complete chain of solutions. Technical Report HPL-2004-189, Hewlett Packard Laboratories, 2004.

[20] B. Zhang, J. Ward, and Q. Feng. Simultaneous parametric maximum flow algorithm with vertex balancing. Technical Report HPL-2005-121, Hewlett Packard Laboratories, 2005.

